# The Generalized FITC Approximation

**Andrew Naish-Guzman & Sean Holden**
Computer Laboratory
University of Cambridge
Cambridge, CB3 0FD. United Kingdom
{agpn2,sbh11}@cl.cam.ac.uk

## Abstract

We present an efficient generalization of the sparse pseudo-input Gaussian process (SPGP) model developed by Snelson and Ghahramani [1], applying it to binary classification problems. By taking advantage of the SPGP prior covariance structure, we derive a numerically stable algorithm with $\mathcal{O}(NM^2)$ training complexity—asymptotically the same as related sparse methods such as the informative vector machine [2], but which more faithfully represents the posterior. We present experimental results for several benchmark problems showing that in many cases this allows an exceptional degree of sparsity without compromising accuracy. Following [1], we locate pseudo-inputs by gradient ascent on the marginal likelihood, but exhibit occasions when this is likely to fail, for which we suggest alternative solutions.

## 1 Introduction

Gaussian processes are a flexible and popular approach to non-parametric modelling. Their conceptually simple architecture is allied with a sound Bayesian foundation, so that not only does their predictive power rival state-of-the-art discriminative methods such as the support vector machine, but they also have the additional benefit of providing an estimate of variance, giving an error bar for their prediction. However, there is a computational price to pay for this robust framework: the time for training scales as $N^3$ for $N$ data points, and the cost of prediction is $\mathcal{O}(N^2)$ per test case.

Recently, there has been great interest in finding sparse approximations to the full Gaussian process (GP) in order to accelerate training and prediction times respectively to $\mathcal{O}(NM^2)$ and $\mathcal{O}(M^2)$, where $M \ll N$ is the size of an auxiliary set, often a subset of the training data, termed variously the inducing inputs, pseudo-inputs or the active set [3, 4, 5, 2, 6, 7, 1]; in this paper, we use the terms interchangeably. Quiñonero-Candela and Rasmussen [8] demonstrated how many of these schemes are related through different approximations to the joint prior over training and test points. In this paper we consider the "fully independent training conditional" or FITC approximation, which appeared originally in Snelson and Ghahramani [1] as the sparse pseudo-input GP (SPGP).

Restricted to a Gaussian noise model, the FITC approximation is entirely tractable; however, for many problems, the Gaussian assumption is inappropriate. In this paper, we describe an extension for non-Gaussian likelihoods, considering as an example probit noise for binary classification. This is not only a common problem, but our results bear out the intuition that sparse methods are well-suited: many data sets enjoy the property that class label does not fluctuate rapidly in the input space, often allowing large regions to be summarized with very few inducing inputs. Contrast this with regression problems, where higher frequency components in the latent signal demand the pseudo-inputs appear in much higher density.

The informative vector machine (IVM) of Lawrence et al. [2] is another sparse GP method that has been extended to non-Gaussian noise models. It is a subset of data method in which the active set

is grown incrementally from the training data using a fast information gain heuristic to find at each stage the optimal inclusion. When a threshold number of points have been added, the algorithm terminates: only data accumulated into the active set are relevant for prediction; remaining points influence the model only in the weak sense of guiding previous steps of the algorithm. Our method is an improvement in three regards: firstly, the FITC approximation makes use of all the data, yielding for the same active set a closer approximation to the posterior distribution. Secondly, unlike the standard IVM approach, we fit a stable posterior at each iteration, providing more accurate marginal likelihood estimates, and derivatives thereof, to allow more reliable model selection. Finally, we argue with experimental justification that the ability to locate inducing inputs independently of the training data, as compared with the greedy approach that drives the IVM, can be a great advantage in finding the sparsest solutions. We discuss these points and other related work in greater detail in section 6.

The structure of this paper is as follows: in section 2 we describe the FITC approximation; this is followed in section 3 by a detailed description of its representation for a non-Gaussian noise model; section 4 provides a brief account of the procedure for model selection; experimental results appear in section 5, which we discuss in section 6; our concluding remarks are in section 7.

## 2   The FITC approximation

Given a domain $\mathcal{X}$ and covariance function $K(\cdot, \cdot) \in \mathcal{X} \times \mathcal{X} \to \mathbb{R}$, a Gaussian process (GP) over the space of real-valued functions of $\mathcal{X}$ specifies the joint distribution at any finite set $\mathbf{X} \subset \mathcal{X}$:

$$p(\mathbf{f}|\mathbf{X}) = \mathcal{N}(\mathbf{f} \,;\, \mathbf{0} \,,\, \mathbf{K_{ff}}) \,,$$

where the $\mathbf{f} = \{f_n\}_{n=1}^{N}$ are (latent) values associated with each $\mathbf{x}_n \in \mathbf{X}$, and $\mathbf{K_{ff}}$ is the *Gram matrix*, the evaluation of the covariance function at all pairs $(\mathbf{x}_i, \mathbf{x}_j)$. We apply Bayes' rule to obtain the posterior distribution over the $\mathbf{f}$, given the observed $\mathbf{X}$ and $\mathbf{y}$, which with the assumption of i.i.d. Gaussian corrupted observations is also normally distributed. Predictions at $\mathbf{X}_\star$ are made by marginalizing over $\mathbf{f}$ in the (Gaussian) joint $p(\mathbf{f}, \mathbf{f}_\star | \mathbf{X}, \mathbf{y}, \mathbf{X}_\star)$. See [9] for a thorough introduction.

In order to derive the FITC approximation, we follow [8] and introduce a set of $M$ inducing inputs $\bar{\mathbf{X}} = \{\bar{\mathbf{x}}_1, \bar{\mathbf{x}}_2, \dots, \bar{\mathbf{x}}_M\}$ with associated latent values $\mathbf{u}$. By the consistency of GPs, we have

$$p(\mathbf{f}, \mathbf{f}_\star | \mathbf{X}, \mathbf{X}_\star, \bar{\mathbf{X}}) = \int p(\mathbf{f}, \mathbf{f}_\star | \mathbf{u}, \mathbf{X}, \mathbf{X}_\star) p(\mathbf{u}|\bar{\mathbf{X}}) \mathrm{d}\mathbf{u} \approx \int q(\mathbf{f}|\mathbf{u}, \mathbf{X}) q(\mathbf{f}_\star|\mathbf{u}, \bar{\mathbf{X}}) p(\mathbf{u}|\bar{\mathbf{X}}) \mathrm{d}\mathbf{u},$$

where $p(\mathbf{u}|\bar{\mathbf{X}}) = \mathcal{N}(\mathbf{u} \,;\, \mathbf{0} \,,\, \mathbf{K_{uu}})$. In the final expression we make the critical approximation by imposing a conditional independence assumption on the joint prior over training and test cases: communication between them must pass through the bottleneck of the inducing inputs. The FITC approximation follows by letting

$$q(\mathbf{f}|\mathbf{u}, \mathbf{X}) = \mathcal{N}\!\left(\mathbf{f} \,;\, \mathbf{K_{fu}} \mathbf{K_{uu}^{-1}} \mathbf{u} \,,\, \mathrm{diag}\left(\mathbf{K_{ff}} - \mathbf{Q_{ff}}\right)\right), \tag{1}$$

$$q(\mathbf{f}_\star|\mathbf{u}, \mathbf{X}_\star) = \mathcal{N}\!\left(\mathbf{f}_\star \,;\, \mathbf{K_{\star u}} \mathbf{K_{uu}^{-1}} \mathbf{u} \,,\, \mathrm{diag}\left(\mathbf{K_{\star\star}} - \mathbf{Q_{\star\star}}\right)\right), \tag{2}$$

where $\mathbf{Q_{ab}} \doteq \mathbf{K_{au}} \mathbf{K_{uu}^{-1}} \mathbf{K_{ub}}$. Of interest for predictions is the posterior distribution over the inducing inputs; this is most efficiently obtained via Bayes' rule after inferring the distribution over $\mathbf{f}$.[1] Using (1) and marginalizing over the exact prior on $\mathbf{u}$ we obtain the approximate prior on $\mathbf{f}$

$$q(\mathbf{f}|\mathbf{X}) = \int \mathcal{N}\!\left(\mathbf{f} \,;\, \mathbf{K_{fu}} \mathbf{K_{uu}^{-1}} \mathbf{u} \,,\, \mathrm{diag}\left(\mathbf{K_{ff}} - \mathbf{Q_{ff}}\right)\right) \mathcal{N}(\mathbf{u} \,;\, \mathbf{0} \,,\, \mathbf{K_{uu}}) \, \mathrm{d}\mathbf{u}$$

$$= \mathcal{N}(\mathbf{f} \,;\, \mathbf{0} \,,\, \mathbf{Q_{ff}} + \mathrm{diag}\left(\mathbf{K_{ff}} - \mathbf{Q_{ff}}\right)) \,. \tag{3}$$

In the original paper, Snelson and Ghahramani placed the pseudo-inputs randomly and learned their locations by non-linear optimization of the marginal likelihood. We have adopted the idea in this paper, but as emphasized in [8], the FITC approximation is applicable regardless of how the inducing

inputs are obtained, and other schemes for their initialization could equally well be married with our algorithm.

In the case of classification, a sigmoidal function assigns class labels $y_n \in \{\pm 1\}$ with a probability that increases monotonically with the latent $f_n$. We use the probit with bias $\beta$,

$$p(y_n|f_n, \beta) = \sigma(y_n(f_n + \beta)) \doteq \int_{-\infty}^{y_n(f_n+\beta)} \mathcal{N}(z \,;\, 0 \,,\, 1)\,\mathrm{d}z. \tag{4}$$

The posterior distribution $p(\mathbf{f}|\mathbf{X}, \mathbf{y})$ is only tractable for Gaussian likelihoods, hence we must resort to a further approximation, either by generating Monte Carlo samples from it or fitting deterministically a Gaussian approximation. Of the latter methods, expectation propagation is possibly the most accurate (at least for GP classification; see [10]), and it is the approach we follow below.

## 3 Inference

We begin with a very brief account of expectation propagation (EP); for more details, see [11, 12]. Suppose we have an intractable distribution over $\mathbf{f}$ whose unnormalized form factorizes into a product of terms, such as a dense Gaussian prior $t_0(\mathbf{f})$ and a series of independent likelihoods $\{t_n(y_n|f_n)\}_{n=1}^N$. EP constructs the approximate posterior as a product of scaled *site functions* $\tilde{t}_n$. For computational tractability, these sites are usually chosen from an exponential family with natural parameters $\boldsymbol{\theta}$, since in this case their product retains the same functional form as its components. The Gaussian $(\boldsymbol{\mu}, \boldsymbol{\Sigma})$ has a natural parameterization $(\mathbf{b}, \boldsymbol{\Pi}) = (\boldsymbol{\Sigma}^{-1}\boldsymbol{\mu}, -\frac{1}{2}\boldsymbol{\Sigma}^{-1})$. If the prior is of this form, its site function is exact:

$$p(\mathbf{f}|\mathbf{y}) = \frac{1}{Z}t_0(\mathbf{f})\prod_{n=1}^N t_n(y_n|f_n) \approx q(\mathbf{f}; \boldsymbol{\theta}) = t_0(\mathbf{f})\prod_{n=1}^N z_n\tilde{t}_n(f_n; \theta_n), \tag{5}$$

where $Z$ is the marginal likelihood and $z_n$ are the scale parameters. Ideally, we would choose $\boldsymbol{\theta}$ at the global minimum of some divergence measure $d(p\|q)$, but the necessary optimization is usually intractable. EP is an iterative procedure that finds a minimizer of $\mathsf{KL}\big(p(\mathbf{f}|\mathbf{y})\|q(\mathbf{f}; \boldsymbol{\theta})\big)$ on a pointwise basis: at each iteration, we select a new site $n$, and from the product of the *cavity* distribution formed by the current marginal with the omission of that site, and the true likelihood term $t_n$, we obtain the so-called *tilted* distribution $q^n(f_n; \boldsymbol{\theta}^{\backslash n})$. A simpler optimization $\min_{\theta_n} \mathsf{KL}\big(q^n(f_n; \boldsymbol{\theta}^{\backslash n})\|q(f_n; \boldsymbol{\theta})\big)$ then fits only the parameters $\theta_n$: this is equivalent to *moment matching* between the two distributions, with scale $z_n$ chosen to match the zeroth-order moments. After each site update, the moments at the remaining sites are liable to change, and several iterations may be required before convergence.

In the discussion below we omit the moment calculations for the probit model, since they correspond to those of traditional GP classification (for more details, consult [9]). Of greater interest is how the mean and covariance structure of the approximate posterior is preserved. Examining the form of the prior (3), we see the covariance consists of a diagonal component $\mathbf{D}_0$ and a rank-$M$ term $\mathbf{P}_0\mathbf{M}_0\mathbf{P}_0^T$, where $\mathbf{P}_0 = \mathbf{K_{fu}}$ and $\mathbf{M}_0 = \mathbf{K_{uu}^{-1}}$ (zero subscripts refer to these initial values; the matrices are updated during the course of the EP iterations). Since the observations $y_n$ are generated i.i.d., we can expect this decomposition to persist in the posterior.

EP requires efficient operations for marginalization to obtain $p(f_n)$, and for updating the posterior distribution after refining a site, as well as for refreshing the posterior to avoid loss of numerical precision. Decomposing $\mathbf{M} = \mathbf{R}^T\mathbf{R}$ into its Cholesky factor,[2] we represent the posterior covariance $\mathbf{A}$ and mean $\mathbf{h}$ by

$$\mathbf{A} = \mathbf{D} + \mathbf{P}\mathbf{R}^T\mathbf{R}\mathbf{P}^T, \qquad\qquad \mathbf{h} = \boldsymbol{\nu} + \mathbf{P}\boldsymbol{\gamma},$$

$$\mathbf{R}_0 := \mathsf{rot180}\left(\mathsf{chol}\big(\mathsf{rot180}\left(\mathbf{K_{uu}}\right)\big)^T \backslash \mathbf{I}\right).$$

where $\mathbf{D}$ is diagonal, $\boldsymbol{\nu}$ is $N \times 1$ and $\boldsymbol{\gamma}$ is $M \times 1$. Writing $\mathbf{p}_n^T = \mathbf{P}_{(n,\cdot)}$ and $d_n = D_{nn}$,

$$A_{nn} = d_n + \|\mathbf{R}\mathbf{p}_n\| \qquad h_n = \nu_n + \mathbf{p}_n^T\boldsymbol{\gamma}, \qquad \text{obtaining marginals in } \mathcal{O}(M^2).$$

Now consider a change in the precision at site $n$ by $\pi_n$. Define the vector $\mathbf{e}$ of length $N$ such that $e_n = 1$ and all other elements are zero. The new covariance $\mathbf{A}_{\text{new}}$ is obtained by inverting the sum of the old precision matrix and the change in precision. If we let $\mathbf{E} = \mathbf{D}^{-1} + \pi_n\mathbf{e}\mathbf{e}^T$, so that

$$\mathbf{E}^{-1} = \mathbf{D} - \frac{\pi_n d_n^2}{1 + \pi_n d_n}\mathbf{e}\mathbf{e}^T \qquad \text{and} \qquad (\mathbf{D}\mathbf{E}\mathbf{D})^{-1} = \mathbf{D}^{-1} - \frac{\pi_n}{1 + \pi_n d_n}\mathbf{e}\mathbf{e}^T,$$

then from the matrix inversion lemma, $\mathbf{A}^{-1} = \mathbf{D}^{-1} - \mathbf{D}^{-1}\mathbf{P}\mathbf{R}^T(\mathbf{R}\mathbf{P}^T\mathbf{D}^{-1}\mathbf{P}\mathbf{R}^T + \mathbf{I})^{-1}\mathbf{R}\mathbf{P}^T\mathbf{D}^{-1}$, and incorporating the update to site $n$,

$$\mathbf{A}_{\text{new}} = \mathbf{E}^{-1} - \mathbf{E}^{-1}\mathbf{D}^{-1}\mathbf{P}\mathbf{R}^T\left(\mathbf{R}\mathbf{P}^T(\mathbf{D}\mathbf{E}\mathbf{D})^{-1}\mathbf{P}\mathbf{R}^T - \mathbf{I} - \mathbf{R}\mathbf{P}^T\mathbf{D}^{-1}\mathbf{P}\mathbf{R}^T\right)^{-1}\mathbf{R}\mathbf{P}^T\mathbf{D}^{-1}\mathbf{E}^{-1}$$

$$= \mathbf{D}_{\text{new}} + \mathbf{P}_{\text{new}}\mathbf{R}_{\text{new}}^T\mathbf{R}_{\text{new}}\mathbf{P}_{\text{new}}^T,$$

where we expand the inversion to obtain a rank-1 downdate to the Cholesky factor $\mathbf{R}$;[3] in summary

$$\mathbf{D}_{\text{new}} = \mathbf{D} - \frac{\pi_n d_n^2}{1 + \pi_n d_n}\mathbf{e}\mathbf{e}^T \quad \mathcal{O}(1) \text{ update}, \qquad \mathbf{P}_{\text{new}} = \mathbf{P} - \frac{\pi_n d_n}{1 + \pi_n d_n}\mathbf{e}\mathbf{p}_n^T \quad \mathcal{O}(M) \text{ update},$$

$$\mathbf{R}_{\text{new}} = \text{chol}_\downarrow\left(\mathbf{R}^T\left(\mathbf{I} - \mathbf{R}\mathbf{p}_n\frac{\pi_n}{1 + \pi_n A_{nn}}\mathbf{p}_n^T\mathbf{R}^T\right)\mathbf{R}\right) \quad \mathcal{O}(M^2) \text{ update}.$$

If the second site parameter, corresponding to precision times mean, is changed by $b_n$, then

$$\mathbf{A}_{\text{new}}^{-1}\mathbf{h}_{\text{new}} = \mathbf{A}^{-1}\mathbf{h} + b_n\mathbf{e} \implies \mathbf{h}_{\text{new}} = \mathbf{A}_{\text{new}}\left(\mathbf{A}_{\text{new}}^{-1} - \pi_n\mathbf{e}\mathbf{e}^T\right)\mathbf{h} + \mathbf{A}_{\text{new}}b_n\mathbf{e}$$

$$= \boldsymbol{\nu}_{\text{new}} + \mathbf{P}_{\text{new}}\boldsymbol{\gamma}_{\text{new}},$$

where

$$\boldsymbol{\nu}_{\text{new}} = \boldsymbol{\nu} + \frac{(b_n + \pi_n\nu_n)d_n}{1 + \pi_n d_n}\mathbf{e} \quad (\mathcal{O}(1)), \qquad \boldsymbol{\gamma}_{\text{new}} = \boldsymbol{\gamma} + \frac{b_n - \pi_n h_n}{1 + \pi_n d_n}\mathbf{R}_{\text{new}}^T\mathbf{R}_{\text{new}}\mathbf{p}_n \quad (\mathcal{O}(M^2)).$$

It is necessary to refresh the covariance and mean every complete EP cycle to avoid loss of precision.

$$\mathbf{D}_{\text{new}} = (\mathbf{I} + \mathbf{D}_0\mathbf{\Pi})^{-1}\mathbf{D}_0 \qquad (\mathcal{O}(N)), \qquad \mathbf{P}_{\text{new}} = (\mathbf{I} + \mathbf{D}_0\mathbf{\Pi})^{-1}\mathbf{P}_0 \qquad (\mathcal{O}(NM)),$$

$$\mathbf{R}_{\text{new}} = \text{rot180}\left(\text{chol}\left(\text{rot180}(\mathbf{I} + \mathbf{R}_0\mathbf{P}_0^T\mathbf{\Pi}(\mathbf{I} + \mathbf{D}_0\mathbf{\Pi})^{-1}\mathbf{P}_0\mathbf{R}_0^T)\right)^T\right)\backslash\mathbf{R}_0 \quad (\mathcal{O}(NM^2)),$$

where $\mathbf{R}_{\text{new}}$ is obtained being careful to ensure the orientations of the factorizations are not mixed. Finally, the mean is refreshed using

$$\boldsymbol{\nu}_{\text{new}} = \mathbf{D}_{\text{new}}\mathbf{b} \quad \text{in } \mathcal{O}(N), \qquad \boldsymbol{\gamma}_{\text{new}} = \mathbf{R}_{\text{new}}^T\mathbf{R}_{\text{new}}\mathbf{P}_{\text{new}}^T\mathbf{b} \quad \text{in } \mathcal{O}(NM),$$

where we have assumed $\mathbf{h}_0 = \mathbf{0}$.

Reviewing the algorithm above, we see that EP costs are dominated by the $\mathcal{O}(M^2)$ Cholesky downdate at each site inclusion. After visiting each of the $N$ sites, we are advised to perform a full refresh, which is $\mathcal{O}(NM^2)$, together leading to asymptotic complexity of $\mathcal{O}(NM^2)$.

## 3.1 Predictions

To make predictions, we marginalize out $\mathbf{u}$ from (2). Initially, Bayes' theorem is used to find the posterior distribution over $\mathbf{u}$ from the inferred posterior over $\mathbf{f}$:

$$p(\mathbf{u}|\mathbf{f}) \propto p(\mathbf{f}|\mathbf{u})p(\mathbf{u}) = \mathcal{N}(\mathbf{u} \mid \mathbf{R}_0^{-1}\mathbf{c}, \mathbf{R}_0^{-1}\mathbf{C}\mathbf{R}_0^{-T}),$$

$$\text{where} \quad \mathbf{c} = \mathbf{C}\mathbf{R}_0\mathbf{P}_0^T\mathbf{D}_0^{-1}\mathbf{f} \quad \text{and} \quad \mathbf{C}^{-1} = \mathbf{I} + \mathbf{R}_0\mathbf{P}_0^T\mathbf{D}_0^{-1}\mathbf{P}_0\mathbf{R}_0^T.$$

Let our posterior approximation be $q(\mathbf{f}|\mathbf{y}) = \mathcal{N}(\mathbf{f}\,;\,\mathbf{h}\,,\,\mathbf{A})$. Hence

$$p(\mathbf{u}|\mathbf{y}) \approx \int p(\mathbf{u}|\mathbf{f})q(\mathbf{f}|\mathbf{y})\mathrm{d}\mathbf{f} = \mathcal{N}(\mathbf{u}\,|\,\mathbf{R}_0^{-1}\boldsymbol{\mu}, \mathbf{R}_0^{-1}\boldsymbol{\Sigma}\mathbf{R}_0^{-T}),$$

$$\text{where}\quad \boldsymbol{\mu} = \mathbf{C}\mathbf{R}_0\mathbf{P}_0^T\mathbf{D}_0^{-1}\mathbf{h} \quad\text{and}\quad \boldsymbol{\Sigma} = \mathbf{C} + \mathbf{C}\mathbf{R}_0\mathbf{P}_0^T\mathbf{D}_0^{-1}\mathbf{A}\mathbf{D}_0^{-1}\mathbf{P}_0\mathbf{R}_0^T\mathbf{C}.$$

Obtaining these terms is $\mathcal{O}(NM^2)$ if we take advantage of the structure of $\mathbf{A}$; the most stable method is via the Cholesky factorization of $\mathbf{C}^{-1}$, rather than forming the explicit inverse. At $\mathbf{x}_\star$,

$$p(f_\star|\mathbf{x}_\star, \mathbf{y}) = \int p(f_\star|\mathbf{u})p(\mathbf{u}|\mathbf{y})\mathrm{d}\mathbf{u} = \mathcal{N}(f_\star\,|\,\mu_\star, \sigma_\star^2);$$

after precomputations, $\mu_\star = \mathbf{k}_\star^T\mathbf{R}_0^T\boldsymbol{\mu}$ is $\mathcal{O}(M)$, and $\sigma_\star^2 = k_{\star\star} + \mathbf{k}_\star^T\mathbf{R}_0^T\left(\boldsymbol{\Sigma} - \mathbf{I}\right)\mathbf{R}_0\mathbf{k}_\star$ is $\mathcal{O}(M^2)$. In the classification domain, we will usually be interested in

$$p(y_\star|\mathbf{x}_\star, \mathbf{y}) = \int p(y_\star|f_\star)p(f_\star|\mathbf{x}_\star, \mathbf{y})\mathrm{d}f_\star = \sigma\left(\frac{y_\star\mu_\star}{\sqrt{1 + \sigma_\star^2}}\right).$$

## 4   Model selection

EP provides an estimate of the log evidence by matching the 0th-order moments $z_n$ at each inclusion. When our posterior approximation is exponential family, Seeger [12] shows the estimate to be

$$L = \sum_{n=1}^{N} \log C_n + \Phi(\boldsymbol{\theta}^{\mathrm{post}}) - \Phi(\boldsymbol{\theta}^{\mathrm{prior}}), \quad\text{where}\quad \log C_n = \log z_n - \Phi(\boldsymbol{\theta}^{\mathrm{post}}) + \Phi(\boldsymbol{\theta}^{\setminus n}),$$

where $\Phi(\cdot)$ denotes the log partition function and $\boldsymbol{\theta}$ are again the natural parameters, with superscripts indicating prior, posterior and cavity. Of interest for model selection are derivatives of the marginal likelihood with respect to hyperparameters $\{\boldsymbol{\xi}, \bar{\mathbf{X}}, \boldsymbol{\beta}\}$, respectively the kernel parameters, pseudo-input locations, and noise model parameters. When the EP fixed point conditions hold (that is, the moments of the tilted distributions match the marginals up to second order for all sites),

$$\nabla_{\boldsymbol{\theta}^{\mathrm{prior}}}L = \boldsymbol{\eta}^{\mathrm{post}} - \boldsymbol{\eta}^{\mathrm{prior}} \qquad\text{and}\qquad \nabla_{\beta_n}L = \log z_n,$$

where $\boldsymbol{\eta}$ denotes the moment parameters of the exponential family (for the Gaussian, these are $(\boldsymbol{\mu}, \boldsymbol{\Sigma} + \boldsymbol{\mu}\boldsymbol{\mu}^T)$) and $\beta_n$ is a parameter of site $n$ (and does not feature in the prior). Finally, we need derivatives $\nabla_{\boldsymbol{\xi}}\boldsymbol{\theta}^{\mathrm{prior}}$ and $\nabla_{\bar{\mathbf{X}}}\boldsymbol{\theta}^{\mathrm{prior}}$. The long-winded details are omitted, but by careful consideration of the covariance structure, it is again possible to limit the complexity to $\mathcal{O}(NM^2)$.

Since we run EP until convergence, our estimates for the marginal likelihood and its derivatives are accurate, allowing us reliablty to fit a model that maximizes the evidence. This is in contrast to the IVM, in which sites excluded from the active set have parameters clamped to zero, and where those included are not iterated to convergence, such that the necessary fixed point conditions do not hold. A particular problem, suffered also by the similar algorithm in [13], is that derivative calculations must be interleaved with site inclusions, and the latter operation tends to disrupt gradient information gained from the previous step. These complications are all sidestepped in our SPGP implementation.

## 5   Experiments

We conducted tests on a variety of data, including two small sets from [14][4] and the benchmark suite of Rätsch.[5] The dimensionality of these classification problems ranges from two to sixty, and the size of the training sets is of the order of 400 to 1000. Results are presented in table 1. For *crabs* and the Rätsch sets, we average over ten folds of the data; for the *synth* problem, Ripley has already divided the data into training and test partitions. Comparisons are made with the full GP classifier, and the SVM, a widely-used discriminative model which in practice is found to yield relatively sparse solutions; we consider also the IVM, a popular framework for building sparse

Table 1: Test errors and predictive accuracy (smaller is better) for the GP classifier, the support vector machine, the informative vector machine, and the sparse pseudo-input GP classifier.

| Data set | | | GPC | | SVM | | IVM | | | SPGPC | | |
|---|---|---|---|---|---|---|---|---|---|---|---|---|
| name | train:test | dim | err | nlp | err | #sv | err | nlp | $M$ | err | nlp | $M$ |
| *synth* | 250:1000 | 2 | 0.097 | *0.227* | 0.098 | 98 | 0.096 | *0.235* | 150 | **0.087** | *0.234* | 4 |
| *crabs* | 80:120 | 5 | 0.039 | *0.096* | 0.168 | 67 | 0.066 | *0.134* | 60 | **0.043** | *0.105* | 10 |
| *banana* | 400:4900 | 2 | 0.105 | *0.237* | 0.106 | 151 | **0.105** | *0.242* | 200 | 0.107 | *0.261* | 20 |
| *breast-cancer* | 200:77 | 9 | 0.288 | *0.558* | **0.277** | 122 | 0.307 | *0.691* | 120 | 0.281 | *0.557* | 2 |
| *diabetes* | 468:300 | 8 | 0.231 | *0.475* | **0.226** | 271 | 0.230 | *0.486* | 400 | 0.230 | *0.485* | 2 |
| *flare-solar* | 666:400 | 9 | 0.346 | *0.570* | **0.331** | 556 | 0.340 | *0.628* | 550 | 0.338 | *0.569* | 3 |
| *german* | 700:300 | 20 | 0.230 | *0.482* | 0.247 | 461 | 0.290 | *0.658* | 450 | **0.236** | *0.491* | 4 |
| *heart* | 170:100 | 13 | 0.178 | *0.423* | **0.166** | 92 | 0.203 | *0.455* | 120 | 0.172 | *0.414* | 2 |
| *image* | 1300:1010 | 18 | 0.027 | *0.078* | 0.040 | 462 | **0.028** | *0.082* | 400 | 0.031 | *0.087* | 200 |
| *ringnorm* | 400:7000 | 20 | 0.016 | *0.071* | 0.016 | 157 | 0.016 | *0.101* | 100 | **0.014** | *0.089* | 2 |
| *splice* | 1000:2175 | 60 | 0.115 | *0.281* | **0.102** | 698 | 0.225 | *0.403* | 700 | 0.126 | *0.306* | 200 |
| *thyroid* | 140:75 | 5 | 0.043 | *0.093* | 0.056 | 61 | 0.041 | *0.120* | 40 | **0.037** | *0.128* | 6 |
| *titanic* | 150:2051 | 3 | 0.221 | *0.514* | **0.223** | 118 | 0.242 | *0.578* | 100 | 0.231 | *0.520* | 2 |
| *twonorm* | 400:7000 | 20 | 0.031 | *0.085* | 0.027 | 220 | 0.031 | *0.085* | 300 | **0.026** | *0.086* | 2 |
| *waveform* | 400:4600 | 21 | 0.100 | *0.229* | 0.107 | 148 | 0.100 | *0.232* | 250 | **0.099** | *0.228* | 10 |

linear models. In all cases, we employed the isotropic squared exponential kernel, avoiding here the anisotropic version primarily to allow comparison with the SVM: lacking a probabilistic foundation, its kernel parameters and regularization constant must be set by cross-validation. For the IVM, hyperparameter optimization is interleaved with active set selection as described in [2], while for the other GP models, we fit hyperparameters by gradient ascent on the estimated marginal likelihood, limiting the process to twenty conjugate gradient iterations; we retained for testing that of three to five randomly initialized models which the evidence most favoured. Results on the Rätsch data for the semi-parametric radial basis function network are omitted for lack of space, but available at the site given in footnote 5. In comparison with that model, SPGP tends to give sparser and more accurate results (with the benefit of a sound Bayesian framework).

Identical tests were run for a range of active set sizes on the IVM and SPGP classifier, and we have attempted to present the large body of results in its most comprehensible form: we list only the *sparsest* competitive solution obtained. This means that using $M$ smaller than shown tends to cause a deterioriation in performance, but not that there is no advantage in increasing the value. After all, as $M \to N$ we expect error rates to match those of the full model (at least for the IVM, which uses a subset of the training data).[6] However, we believe that in exploring the behaviour of a sparse model, the essential question is: what is the greatest sparsity we can achieve without compromising performance? (since if sparsity were not an issue, we would simply revert to the original GP). Small values of $M$ for the FITC approximation were found to give remarkably low error rates, and incremented singly would often give an improved approximation. In contrast, the IVM predictions were no better than random guesses for even moderate $M$—it usually failed if the active set was smaller than a threshold around $N/3$, where it was simply discarding too much information—and greater step sizes were required for noticeable improvements in performance. With a few exceptions then, for FITC we explored small $M$, while for the IVM we used larger values, more widely spread.

More challenging is the task of discriminating 4s from non-4s in the USPS digit database: the data are 256-dimensional, and there are 7291 training and 2007 test points. With 200 pseudo-inputs (and 51,200 parameters for optimization), error rates for SPGPC are 1.94%, with an average negative log probability of 0.051 nats. These figures improve when the allocation is raised to 400 pseudo-inputs, to 1.79% and 0.048 nats. When provided with only 200 points, the IVM figures are 9.97% and 0.421 nats—this can be regarded as a failure to generalize, since it corresponds to labelling all test inputs as "not 4"—but given an active set of 400 it reaches error rates of 1.54% and NLP of 0.085 nats.

# 6  Discussion

A sparse approximation closely related to FITC is the "deterministic training conditional" (DTC), whose covariance consists solely of the low-rank term $\mathbf{LML}^T$; it has appeared elsewhere under the name *projected latent variables* [13]. In generative terms, DTC first obtains a posterior process by conditioning on the inducing inputs; observations $\mathbf{y}$ are then drawn as noisy samples of the mean of this process. FITC is similar, but the draws are noisy samples from the posterior process itself—hence, while the noise component for DTC is a constant corruption $\sigma^2$, for FITC it grows away from the inducing inputs to $K_{nn} + \sigma^2$. In comparing their SPGP model with DTC, Snelson and Ghahramani [1] suggest that it is for this reason (i.e. due to the diagonal component in the covariance in FITC) that the optimization of pseudo-inputs by gradient ascent on the marginal likelihood can succeed: without the noise reduction afforded locally by relocating pseudo-inputs, DTC does not provide a sufficiently large gradient for them to move, and the optimization gets stuck. We believe the same mechanism operates in general for non-Gaussian noise.

This difficulty would not be significant if alternative heuristics for building the active set greedily were effective. We hypothesize however that the most informative vectors in the greedy sense of the IVM tend to be those which lie close to the decision boundary. Such points will have a relatively strong influence on its shape since the effect of the kernel falls off exponentially in distance squared. A preferable solution may be that empirically found to occur with Tipping's relevance vector machine (RVM) [15], a degenerate GP where a particular prior on weights means only a few basis functions survive an evidence maximization procedure to form the model;[7] there, the classifier was often parameterized by points distant from the decision boundary, suggested to be more "representative" of the data.

We illustrate with a simple example that, provided the optimization is feasible, very sparse solutions may more easily be found if the inducing inputs can be positioned independently of the data. This allows the size of the active set to grow with the complexity of the problem, rather than with $N$, the number of training points. We drew samples from a two-dimensional "xor" problem, consisting of four unit-variance Gaussian clusters at $(\pm 1.5, \pm 1.5)$ with a small overlap, giving an optimal error rate of around 13% and in loose terms a complexity which requires an active set of size four. By increasing the size of the training set $N$ in increments from 40 to 400, we obtained the learning curves of figure 1 for the IVM and FITC models: plotted against $N$ is the size of active set required for the error rate to fall below 15%. Whereas the FITC model requires a constant four points to explain the data, the demands of the IVM appear to increase almost linearly with $N$.

Evidently, the FITC model is able to capture salient details more readily than the IVM, but we may object that it is also a richer likelihood. We therefore show learning curves for the FITC approximation run using the IVM active set and, generously, optimal kernel parameters. With a relatively simple and low-dimensional problem, the benefit of the adaptable active set that FITC offers is clearly less significant than that of the improved approximation itself—although there is a factor of 2 difference, and we believe the effects will be more pronounced for more complex data. However, a sensible compromise where optimization of all pseudo-inputs is computationally infeasible is to run the IVM to obtain an initial active set, but then switch to the FITC approximation and optimize only kernel parameters, or just a small selection of the pseudo-inputs. Another option, explored by Snelson and Ghaharamani [17] for this model in the case of regression, is to learn a low dimensional projection of the data—advantageous, since in this setting the pseudo-inputs only operate under projection and can be treated as low-dimensional, potentially reducing significantly the scale of the optimization problem. We report results of this extension in future work.

# 7  Conclusions

We have presented an efficient and numerically stable way of implementing the sparse FITC model in Gaussian processes. By way of example we considered binary classification in which extra data points are introduced to form a continuously adaptable active set. We have demonstrated that the locations of these pseudo-inputs can be fit synchronously with parameters of the kernel, and that

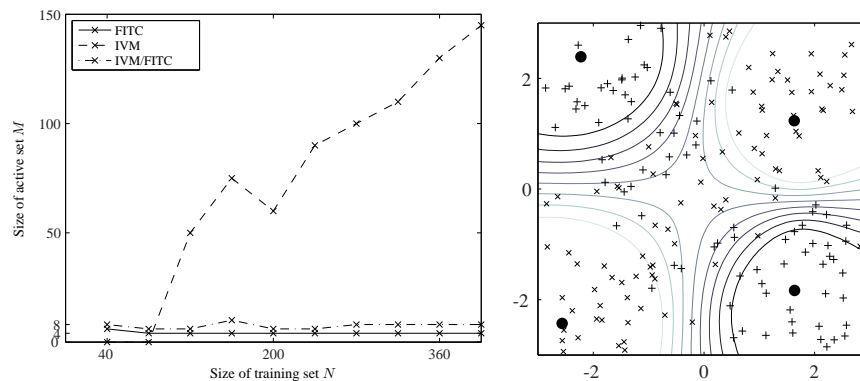

Figure 1: Left: learning curves for the toy problem described in the text. Right: contours of posterior probability for FITC in ten CG iterations from a random initialization of pseudo-inputs (black dots).

this procedure allows for very sparse solutions. Certain data sets, particularly those of very high dimensionality, are not amenable to this approach since the number of hyperparameters is unfeasibly large for non-linear optimization. In this case, we suggest resorting to a greedy approach, using a fast heuristic like the IVM to build the active set, but adopting the FITC approximation thereafter. An alternative which deserves investigation is to attempt an initial round of k-means clustering.

## Footnotes

[1]We could also infer the posterior over $\mathbf{u}$ directly, rather than marginalizing over the inducing inputs as here. Running EP in this setting, each site maintains a belief about the full $M \times M$ covariance, and we obtain a slower $\mathcal{O}(NM^3)$ algorithm. Furthermore, calculations to evaluate the derivatives of the log marginal likelihood with respect to inducing inputs $\bar{\mathbf{x}}_m$ are significantly complicated by their presence in both prior and likelihood.

[2]Care must be taken that the factors share the correct orientation. When our environment offers only upper Cholesky factors $\mathbf{R}^T\mathbf{R}$, the initialization of $\mathbf{R}_0 = \mathsf{chol}\left(\mathbf{K_{uu}^{-1}}\right)$ can be achieved without computing the explicit inverse via the following matrix rotations:

[3]If the factor $\frac{\pi_n}{1 + \pi_n A_{nn}}$ is negative, we make a rank-1 update, guaranteed to preserve the positive definite property. Note that on rare occasions, loss of precision can cause a downdate to result in a non-positive definite covariance matrix. If this occurs, we should abort the update and refresh the posterior from scratch. In any case, to improve conditioning, it is recommended to add a small multiple of the identity to the prior $\mathbf{M}_0$.

[4]Available from `http://www.stats.ox.ac.uk/pub/PRNN/`.

[5]Available from `http://ida.first.fhg.de/projects/bench/benchmarks.htm`.

[6]Note that the evidence is a poor metric for choosing $M$ since it tends to increase monotonically as the explicative power of the full GP is restored.

[7]We have not compared our model with the RVM since that approximation suffers from nonsensical variance estimates away from the data. Rasmussen and Quiñonero-Candela [16] show how it can be "healed" through augmentation, but the resulting model is no longer sparse in the sense of providing $\mathcal{O}(M^2)$ predictions.

## References

[1] Edward Snelson and Zoubin Ghahramani. Sparse Gaussian processes using pseudo-inputs. In *Advances in Neural Information Processing Systems 18*. MIT Press, 2005.

[2] Neil Lawrence, Matthias Seeger, and Ralf Herbrich. Fast sparse Gaussian process methods: the informative vector machine. In *Advances in Neural Information Processing Systems 15*. MIT Press, 2003.

[3] Manfred Opper and Ole Winther. Gaussian processes for classification: mean field methods. *Neural Computation*, 12(11):2655–2684, 2000.

[4] Volker Tresp. A Bayesian committee machine. *Neural Computation*, 12(11):2719–2741, 2000.

[5] Alex Smola and Peter Bartlett. Sparse greedy Gaussian process regression. In *Advances in Neural Information Processing Systems 13*. MIT Press, 2001.

[6] Lehel Csató. *Gaussian processes: iterative sparse approximations*. PhD thesis, Aston University, 2002.

[7] Matthias Seeger. *Bayesian Gaussian process models: PAC-Bayesian generalisation error bounds and sparse approximations*. PhD thesis, University of Edinburgh, 2003.

[8] Joaquin Quiñonero-Candela and Carl Edward Rasmussen. A unifying view of sparse approximate Gaussian process regression. *Journal of Machine Learning Research*, 6(12):1939–1959, 2005.

[9] Carl Rasmussen and Christopher Williams. *Gaussian processes for machine learning*. MIT Press, 2006.

[10] Malte Kuss and Carl Edward Rasmussen. Assessing approximations for Gaussian process classification. In *Advances in Neural Information Processing Systems 18*. MIT Press, 2005.

[11] Thomas Minka. *A family of algorithms for approximate Bayesian inference*. PhD thesis, Massachusetts Institute of Technology, 2001.

[12] Matthias Seeger. Expectation propagation for exponential families, 2005. Available from http://www.cs.berkeley.edu/~mseeger/papers/epexpfam.ps.gz.

[13] Matthias Seeger, Christopher Williams, and Neil Lawrence. Fast forward selection to speed up sparse Gaussian process regression. In *Proceedings of the 9th International Workshop on AI Stats*. Society for Artificial Intelligence and Statistics, 2003.

[14] Brian Ripley. *Pattern recognition and neural networks*. Cambridge University Press, 1996.

[15] Michael E. Tipping. Sparse Bayesian learning and the relevance vector machine. *Journal of Machine Learning Research*, 1:211–244, 2001.

[16] Carl Edward Rasmussen and Joaquin Quiñonero-Candela. Healing the relevance vector machine through augmentation. In *Proceedings of 22nd ICML*. ACM Press, 2005.

[17] Edward Snelson and Zoubin Ghahramani. Variable noise and dimensionality reduction for sparse Gaussian processes. In *Proceedings of the 22nd Annual Conference on Uncertainty in AI*. AUAI Press, 2006.

